# Node Splitting: A Constructive Algorithm for Feed-Forward Neural Networks

**Mike Wynne-Jones**
Research Initiative in Pattern Recognition
St. Andrews Road, Great Malvern
WR14 3PS, UK
mikewj@hermes.mod.uk

## Abstract

A constructive algorithm is proposed for feed-forward neural networks, which uses node-splitting in the hidden layers to build large networks from smaller ones. The small network forms an approximate model of a set of training data, and the split creates a larger more powerful network which is initialised with the approximate solution already found. The insufficiency of the smaller network in modelling the system which generated the data leads to oscillation in those hidden nodes whose weight vectors cover regions in the input space where more detail is required in the model. These nodes are identified and split in two using principal component analysis, allowing the new nodes to cover the two main modes of each oscillating vector. Nodes are selected for splitting using principal component analysis on the oscillating weight vectors, or by examining the Hessian matrix of second derivatives of the network error with respect to the weights. The second derivative method can also be applied to the input layer, where it provides a useful indication of the relative importances of parameters for the classification task. Node splitting in a standard Multi Layer Perceptron is equivalent to introducing a hinge in the decision boundary to allow more detail to be learned. Initial results were promising, but further evaluation indicates that the long range effects of decision boundaries cause the new nodes to slip back to the old node position, and nothing is gained. This problem does not occur in networks of localised receptive fields such as radial basis functions or gaussian mixtures, where the technique appears to work well.

# 1   Introduction

To achieve good generalisation in neural networks and other techniques for inferring a model from data, we aim to match the number of degrees of freedom of the model to that of the system generating the data. With too small a model we learn an incomplete solution, while too many free parameters capture individual training samples and noise.

Since the optimum size of network is seldom known in advance, there are two alternative ways of finding it. The *constructive algorithm* aims to build an approximate model, and then add new nodes to learn more detail, thereby approaching the optimum network size from below. *Pruning* algorithms, on the other hand, start with a network which is known to be too big, and then cut out nodes or weights which do not contribute to the model. A review of recent techniques [WJ91a] has led the author to favour the constructive approach, since pruning still requires an estimate of the optimum size, and the initial large networks can take a long time to train. Constructive algorithms offer fast training of the initial small networks, with the network size and training slowness reflecting the amount of information already learned. The best approach of all would be a constructive algorithm which also allowed the pruning of unnecessary nodes or weights from the network.

The constructive algorithm trains a network until no further detail of the training data can be learned, and then adds new nodes to the network. New nodes can be added with random weights, or with pre-determined weights. Random weights are likely to disrupt the approximate solution already found, and are unlikely to be initially placed in parts of the weight space where they can learn something useful, although encouraging results have been reported in this area.[Ash89] This problem is likely to be accentuated in higher dimensional spaces. Alternatively, weights can be pre-determined by measurements on the performance of the seed network, and this is the approach adopted here. One node is turned into two, each with half the output weight. A divergence is introduced in the weights into the nodes which is sufficient for them behave independently in future training without disrupting the approximate solution already found.

# 2   Node-Splitting

A network is trained using standard techniques until no further improvement on training set performance is achieved. Since we begin with a small network, we have an approximate model of the data, which captures the dominant properties of the generating system but lacks detail. We now freeze the weights in the network, and calculate the updates which would be made them, using simple gradient descent, by each separate training pattern. Figure 1 shows the frozen vector of weights into a single hidden node, and the scatter of proposed updates around the equilibrium position.

The picture shows the case of a hidden node where there is one clear direction of oscillation. This might be caused by two clusters of data within a class, each trying to use the node in its own area of the input space, or by a decision boundary pulled clockwise by some patterns and anticlockwise by others. If the oscillation is strong, either in its exhibition of a clear direction or in comparison with other

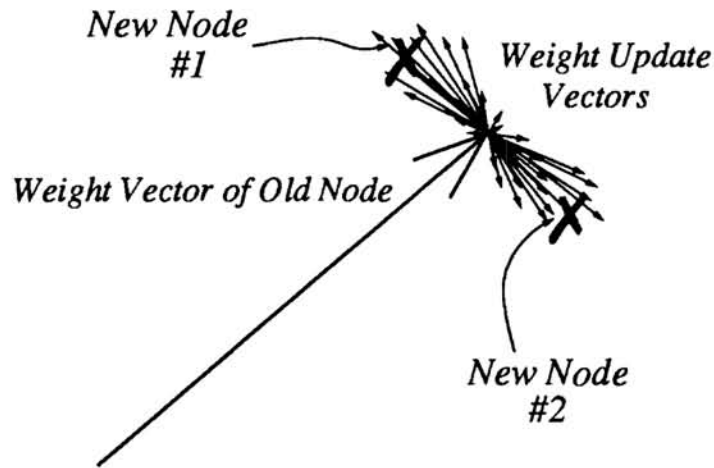

Figure 1: A hidden node weight vector and updates proposed by individual training patterns

nodes in the same layer, then the node is split in two. The new nodes are placed one standard deviation either side of the old position. While this divergence gives the nodes a push in the right direction, allowing them to continue to diverge in later training, the overall effect on the network is small. In most cases there is very little degradation in performance as a result of the split.

The direction and size of oscillation are calculated by principal component analysis of the weight updates. By a traditional method, we are required to make a covariance matrix of the weight updates for the weight vector into each node:

$$\mathbf{C} = \sum_p \delta\mathbf{w}\delta\mathbf{w}^T \tag{1}$$

where p is the number of patterns. The matrix is then decomposed to a set of eigenvalues and eigenvectors; the largest eigenvalue is the variance of oscillation and the corresponding eigenvector is its direction. Suitable techniques for performing this decomposition include Singular Value Decomposition and Householder Reduction. [Vet86] A much more suitable way of calculating the principal components of a stream of continuous measurements such as weight updates is iterative estimation. An estimate is stored for each required principal component vector, and the estimates are updated using each sample. [Oja83, San89] By Oja's method, the scalar product of the current sample vector with each current estimate of the eigenvectors is used as a matching coefficient, $M$. The matching coefficient is used to re-estimate the eigenvalues and eigenvectors, in conjunction with a gain term $\lambda$ which decays as the number of patterns seen increases. The eigenvectors are updated by a proportion $\lambda M$ of the current sample, and the eigenvalues by $\lambda M^2$. The trace (sum of eigenvalues) can also be estimated simply as the mean of the traces (sum of diagonal elements) of the individual sample covariance matrices. The principal component vectors are renormalised and orthogonalised after every few updates. This algorithm is of order $n$, the number of eigenvalues required, for the re-estimation, and $O(n^2)$ for the orthogonalisation; the matrix decomposition method can take exponential

time, and is always much slower in practice.

In a recent paper on *Meiosis Networks*, Hanson introduced stochastic weights in the multi layer perceptron, with the aim of avoiding local minima in training.[Han90] A sample was taken from a gaussian distribution each time a weight was used; the mean was updated by gradient descent, and the variance reflected the network convergence. The variance was allowed to decay with time, so that the network would approach a deterministic state, but was increased in proportion to the updates made to the mean. While the network was far from convergence these updates were large, and the variance remained large. Node splitting was implemented in this system, in nodes where the variances on the weights were large compared with the means. In such cases, two new nodes were created with the weights one standard deviation either side of the old mean: one SD is added to all weights to one node, and subtracted for all weights to the other. Preliminary results were promising, but there appear to be two problems with this approach for node-splitting. First, the splitting criterion is not good: a useless node with all weights close to zero could have comparatively large variances on the weights owing to noise. This node would be split indefinitely. Secondly and more interestingly, the split is made without regard to the correlations in sign between the weight updates, shown as dots in the scatter plots of figure 2. In figure 2a, Meiosis would correctly place new nodes in the positions marked with crosses, while in figure 2b, the new nodes would be placed in completely the wrong places. This problem does not occur in the node splitting scheme based on principal component analysis.

(a)                                         (b)

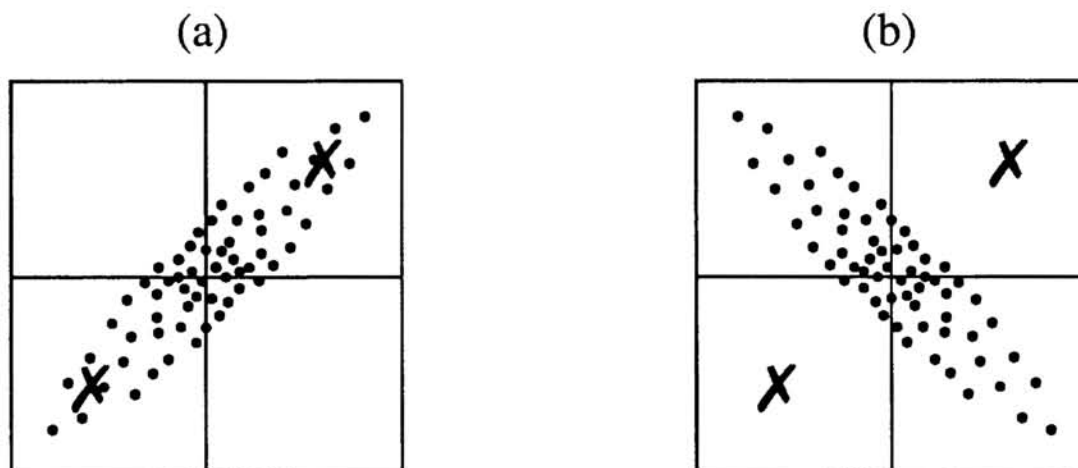

Figure 2: Meiosis networks split correctly if the weight updates are correlated in sign (a), but fail when they are not (b).

## 3  Selecting nodes for splitting

Node splitting is carried out in the direction of maximum variance of the scatter plot of weight updates proposed by individual training samples. The hidden layer nodes most likely to benefit from splitting are those for which the non-spherical nature

of the scatter plot is most pronounced. In later implementations this criterion was measured by comparing the largest eigenvalue with the sum of the eigenvalues, both these quantities being calculated by the iterative method. This is less simple in cases where there are a number of dominant directions of variance; the scatter plot might, for example be a four dimensional disk in a ten dimensional space, and hence present the possibility of splitting one node into eight. It is hoped that these more complicated splits will be the subject of further research.

An alternative approach in determining the need of nodes to be split, in comparison with other nodes in the same layer, is to use the second derivatives of the network error with respect to a parameter of the nodes which is normalised across all nodes in a given layer of the network. Such a parameter was proposed by Mozer and Smolensky in [Smo89]: a multiplicative gating function is applied to the outputs of the nodes, with its gating parameter set to one. Small increments in this parameter can be used to characterise the error surface around the unity value, with the result that derivatives are normalised across all nodes in a given layer of the network. Mozer and Smolensky replaced the sum squared error criterion with a modulus error criterion to preserve non-zero gradients close to the local minimum reached in training; we prefer to characterise the true error surface by means of second derivatives, which can be calculated by repeated use of the chain rule (backpropagation). Backpropagation of second derivatives has previously been reported in [Sol90] and [Hea90].

Since a high curvature error minimum in the space of the gating parameter for a particular node indicates steep gradients surrounding the minimum, it is these nodes which exhibit the greatest instability in their weight-space position. In the weight space, if the curvature is high only in certain directions, we have the situation in figure 1, where the node is oscillating, and is in need of splitting. If the curvature is high in all directions in comparison with other nodes, the network is highly sensitive to changes in the node or its weights, and again it will benefit from splitting.

At the other end of the scale of curvature sensitivity, a node or weight with very low curvature is one to which the network error is quite insensitive, and the parameter is a suitable candidate for pruning. This scheme has previously been used for weight pruning by Le Cun, Denker et al. [Sol90], and offers the potential for an integrated system of splitting and pruning - a truly adaptive network architecture.

## 3.1    Applying the sensitivity measure to input nodes

In addition to using the gating parameter sensitivity to select nodes for pruning, Mozer and Smolensky mention the possibility of using it on the input nodes to indicate those inputs to which the classification is most sensitive. This has been implemented in our system with the second derivative sensitivity measure, and applied to a large financial classification problem supplied by THORN EMI Research. The analysis was carried out on the 78-dimensional data, and the input sensitivities varied over several orders of magnitude. The inputs were grouped into four sets according to sensitivity, and MLPs of 10 hidden nodes were trained on each subset of the data. While the low sensitivity groups failed to learn anything at all, the higher sensitivity groups quickly attained a reasonable classification rate. Identification of useless inputs leads to greatly increased training speed in future analysis, and can

yield valuable economies in future data collection. This work is reported in more detail in [WJ91b].

## 4    Evaluation in Multi Layer Perceptron networks

Despite the promising results from initial evaluations, further testing showed that the splitter technique was often unable to improve on the performance of the network used as a seed for the first split. These test were carried out on a number of different classification problems, where large numbers of hidden nodes were already known to be required, and with a number of different splitting criteria. Prolonged experimentation and consideration of this failure lead to the hypothesis that a split might be made to correct some misclassified patterns in one region of the input space but, owing to the long range effects of MLP decision boundaries, the changed positions of the planes might cause a much greater number of misclassifications elsewhere. These would tend to cause the newly created nodes to slip back to the position of the node from which they were created, with no overall benefit. This possibility was tested by re-implementing the splitter technique in a gaussian mixture modeling system, which uses a network of localised receptive fields, and hence does not have the long range effects which occurred in the multi layer perceptron.

## 5    Implementation of the splitter in a Gaussian Mixture Model, and the results

The Gaussian Mixtures Model [Cox91] is a clustering algorithm, which attempts to model the distribution of a points in a data set. It consists of a number of multivariate gaussian distributions in different positions in the input space, and with different variances in different directions. The responses of these receptive fields (bumps) are weighted and summed together; the weights are calculated to satisfy the PDF constraint that the responses should sum to one over the data set. For the experiments on node splitting, the variance was the same in all directions for a particular bump, leading to a model which is a sum of weighted spherical gaussian distributions of different sizes and in different positions. The model is trained by gradient ascent in the likelihood of the model fitting the data, which leads to a set of learning rules for re-estimating the weights, then the centre positions of the receptive fields, then their variances.

For the splitter, a small model is trained until nothing more can be learned, and the parameters are frozen. The training set is run through once more, and the updates are calculated which each pattern attempts to make to the centre position of each receptive field. The first principal component and trace of these updates are calculated by the iterative method, and any nodes for which the principal component variance is a large proportion of the trace is split in two.

The algorithm is quick to converge, and is slowed down only a little by the overhead of computing the principal component and trace. Figure 3 shows the application of the gaussian mixture splitter to modelling a circle and an enclosing annulus; in the circle (a) there is no dominant principal component direction in the data covered by the receptive field of each node (shown at one standard deviation by a circle), while

in (b) three nodes are clearly insufficient to model the annulus, and one has just undergone a split. (c) shows the same data set and model a little later in training after a number of splits have taken place. The technique has been evaluated on a number of other simple problems, with no negative results to date.

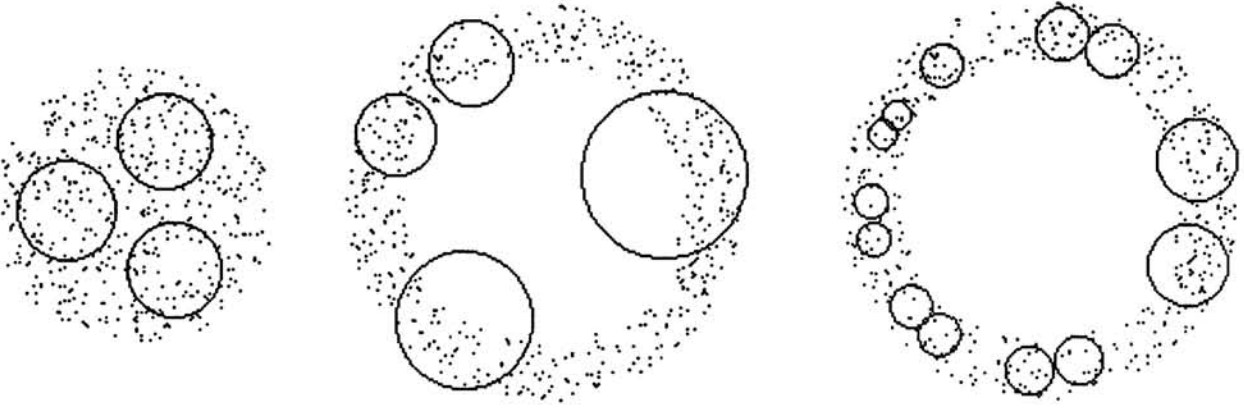

Figure 3: Gaussian mixture model with node-splitting applied to a circle and surrounding annulus

## 6    Conclusions

The splitter technique based on taking the principal component of the influences on hidden nodes in a network, has been shown to be useful in the multi layer perceptron in only a very limited number of cases. The split in this kind of network corresponds to a hinge in the decision boundary, which corrects the errors for which it was calculated, but usually caused for more errors in other parts of the input space. This problem does not occur in networks of localised receptive fields such as radial basis functions of gaussian mixture distributions, where it appears to work very well. Further studies will include splitting nodes into more than two, in cases where there is more than one dominant principal component, and applying node-splitting to different modelling algorithms, and to gaussian mixtures in hidden markov models for speech recognition.

The analysis of the sensitivity of the network error to individual nodes gives an ordered list which can be used for both splitting and pruning in the same network, although splitting does not generally work in the MLP. This measure has been demonstrated in the input layer, to identify which network inputs are more or less useful in the classification task.

### Acknowledgements

The author is greatly indebted to John Bridle and Steve Luttrell of RSRE, Neil Thacker of Sheffield University, and colleagues in the Research Initiative in Pattern

Recognition and its member companies for helpful comments and advice; also to David Bounds of Aston University and RIPR for advice and encouragement.

## References

[Ash89]   Timur Ash. Dynamic node creation in backpropagation networks. Technical Report 8901, Institute for Cognitive Science, UCSD, La Jolla, California 92093, February 1989.

[Cox91]   John S Bridle & Stephen J Cox. Recnorm: Simultaneous normalisation and classification applied to speech recognition. In Richard P Lippmann & John E Moody & David S Touretzky, editor, *Advances in Neural Information Processing Systems 3*, pages 234–240, San Mateo, CA, September 1991. Morgan Kaufmann Publishers.

[Han90]   Stephen José Hanson. Meiosis networks. In David S Touretzky, editor, *Advances in Neural Information Processing Systems 2*, pages 533–541, San Mateo, CA, April 1990. Morgan Kaufmann Publishers.

[Hea90]   Anthony JR Heading. An analysis of noise tolerance in multi-layer perceptrons. Research Note SP4 122, Royal Signals and Radar Establishment, St Andrews Road, Malvern, Worcestershire, WR14 3PS, UK, July 1990.

[Oja83]   E Oja. *Subspace Methods of Pattern Recognition*. Research Studies Press Ltd, Letchworth, UK, 1983.

[San89]   TD Sanger. Optimal unsupervised learningin a single-layer linear feedforward neural network. *Neural Networks*, 2:459–473, 1989.

[Smo89]   MC Mozer & P Smolensky. Skeletonization: A technique for trimming the fat from a neural network. In DS Touretzky, editor, *Advances in Neural Information Processing Systems 1*, pages 107–115, San Mateo, CA, April 1989. Morgan Kaufmann Publishers.

[Sol90]   Yann Le Cun & John S Denker & Sara A Solla. Optimal brain damage. In David S Touretzky, editor, *Advances in Neural Information Processing Systems 2*, pages 598–605, San Mateo, CA, April 1990. Morgan Kaufmann Publishers.

[Vet86]   WH Press & BP Flannery & SA Teukolsky & WT Vetterling. *Numerical Recipes in C: The Art of Scientific Computing*. Cambrigde University Press, 1986.

[WJ91a]   Mike Wynne-Jones. Constructive algorithms and pruning: Improving the multi layer perceptron. In R Vichnevetsky & JJH Miller, editor, *Proceedings of the 13th IMACS World Congress on Computation and Applied Mathematics*, pages 747–750, Dublin, July 1991. IMACS '91, IMACS.

[WJ91b]   Mike Wynne-Jones. Self-configuring neural networks, a new constructive algorithm, and assessing the importance of individual inputs. Technical Report X2345/1, Thorn EMI Central Research Laboratories, Dawley Road, Hayes, Middlesex, UB3 1HH, UK, March 1991.
